# The Infinite Factorial Hidden Markov Model

**Jurgen Van Gael**[*]
Department of Engineering
University of Cambridge, UK
jv279@cam.ac.uk

**Yee Whye Teh**
Gatsby Unit
University College London, UK
ywteh@gatsby.ucl.ac.uk

**Zoubin Ghahramani**
Department of Engineering
University of Cambridge, UK
zoubin@eng.cam.ac.uk

## Abstract

We introduce a new probability distribution over a potentially infinite number of binary Markov chains which we call the Markov Indian buffet process. This process extends the IBP to allow temporal dependencies in the hidden variables. We use this stochastic process to build a nonparametric extension of the factorial hidden Markov model. After constructing an inference scheme which combines slice sampling and dynamic programming we demonstrate how the infinite factorial hidden Markov model can be used for blind source separation.

## 1  Introduction

When modeling discrete time series data, the *hidden Markov model* [1] (HMM) is one of the most widely used and successful tools. The HMM defines a probability distribution over observations $y_1, y_2, \cdots y_T$ using the following generative model: it assumes there is a hidden Markov chain $s_1, s_2, \cdots, s_T$ with $s_t \in \{1 \cdots K\}$ whose dynamics is governed by a $K$ by $K$ stochastic transition matrix $\boldsymbol{\pi}$. At each timestep $t$, the Markov chain generates an output $y_t$ using some likelihood model $F$ parametrized by a state dependent parameter $\theta_{s_t}$. We can write the probability distribution induced by the HMM as follows[1]

$$p(y_{1:T}, s_{1:T}) = \prod_{t=1}^{T} p(s_t|s_{t-1})p(y_t|s_t) = \prod_{t=1}^{T} \pi_{s_{t-1},s_t} F(y_t; \theta_{s_t}). \tag{1}$$

Figure 1 shows the graphical model for the HMM.

One shortcoming of the hidden Markov model is the limited representational power of the latent variables. One way to look at the distribution defined by the HMM is to write down the marginal distribution of $y_t$ given the previous latent state $s_{t-1}$

$$p(y_t|s_{t-1}) = \sum_{s_t} p(s_t|s_{t-1})p(y_t|s_t) = \sum_{s_t} \pi_{s_{t-1},s_t} F(y_t; \theta_{s_t}). \tag{2}$$

Equation (2) illustrates that the observations are generated from a dynamic mixture model. The *factorial hidden Markov model* (FHMM), developed in [2], addresses the limited representational power of the hidden Markov model. The FHMM extends the HMM by representing the hidden state

---

[*]http://mlg.eng.cam.ac.uk/jurgen

[1]To make the notation more convenient, we assume w.l.o.g. that for all our models, all latent chains start in a dummy state that is in the 0 state. E.g. for the HMM $s_0 = 0$, for the FHMM $s_0^{(m)} = 0$ for all $m$.

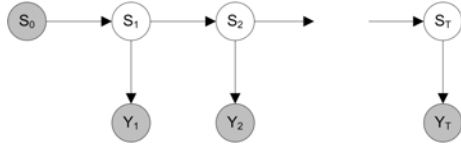

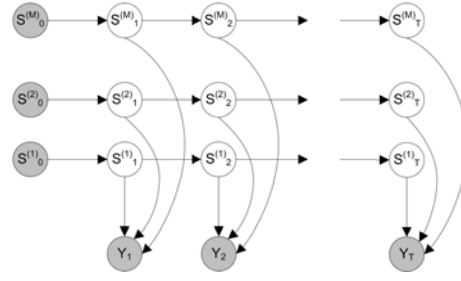

Figure 1: The Hidden Markov Model          Figure 2: The Factorial Hidden Markov Model

in a factored form. This way, information from the past is propagated in a distributed manner through a set of parallel Markov chains. The parallel chains can be viewed as latent features which evolve over time according to Markov dynamics. Formally, the FHMM defines a probability distribution over observations $y_1$ $y_2$ $y_T$ as follows: $M$ latent chains $s^{(1)}$ $s^{(2)}$ $s^{(M)}$ evolve according to Markov dynamics and at each timestep $t$, the Markov chains generate an output $y_t$ using some likelihood model $F$ parameterized by a joint state-dependent parameter $s_t^{(1:m)}$. The graphical model in figure 2 shows how the FHMM is a special case of a dynamic Bayesian network. The FHMM has been successfully applied in vision [3], audio processing [4] and natural language processing [5]. Unfortunately, the dimensionality $M$ of our factorial representation or equivalently, the number of parallel Markov chains, is a new free parameter for the FHMM which we would prefer learning from data rather than specifying it beforehand.

Recently, [6] introduced the basic building block for nonparametric Bayesian factor models called the *Indian Buffet Process* (IBP). The IBP defines a distribution over *infinite binary* matrices $Z$ where element $z_{nk}$ denotes whether datapoint $n$ has feature $k$ or not. The IBP can be combined with distributions over real numbers or integers to make the features useful for practical problems.

In this work, we derive the basic building block for nonparametric Bayesian factor models for time series which we call the *Markov Indian Buffet Process* (mIBP). Using this distribution we build a nonparametric extension of the FHMM which we call the *Infinite Factorial Hidden Markov Model* (iFHMM). This construction allows us to learn a factorial representation for time series.

In the next section, we develop the novel and *generic* nonparametric mIBP distribution. Section 3 describes how to use the mIBP do build the iFHMM. Which in turn can be used to perform independent component analysis on time series data. Section 4 shows results of our application of the iFHMM to a blind source separation problem. Finally, we conclude with a discussion in section 5.

## 2   The Markov Indian Buffet Process

Similar to the IBP, we define a distribution over binary matrices to model whether a feature at time $t$ is on or off. In this representation rows correspond to timesteps and the columns to features or Markov chains. We want the distribution over matrices to satisfy the following two properties: (1) the potential number of columns (representing latent features) should be able to be arbitrary large; (2) the rows (representing timesteps) should evolve according to a Markov process.

Below, we will formally derive the mIBP distribution in two steps: first, we describe a distribution over binary matrices with a finite number of columns. We choose the hyperparameters carefully so we can easily integrate out the parameters of the model. In a second phase, we take the limit as the number of features goes to infinity in a manner analogous to [7]'s derivation of infinite mixtures.

### 2.1   A finite model

Let $S$ represent a binary matrix with $T$ rows (datapoints) and $M$ columns (features). $s_{tm}$ represents the hidden state at time $t$ for Markov chain $m$. Each Markov chain evolves according to the transition matrix

$$W^{(m)} = \begin{matrix} 1 & a_m & a_m \\ 1 & b_m & b_m \end{matrix} \qquad (3)$$

where $W_{ij}^{(m)} = p(s_{t+1,m} = j | s_{tm} = i)$. We give the parameters of $W^{(m)}$ distributions $a_m \sim$ Beta$(\alpha/M, 1)$ and $b_m \sim$ Beta$(\gamma, \delta)$. Each chain starts with a dummy zero state $s_{0m} = 0$. The hidden state sequence for chain $m$ is generated by sampling $T$ steps from a Markov chain with transition matrix $W^{(m)}$. Summarizing, the generative specification for this process is

$$\forall m \in \{1, 2, \cdots, M\} : a_m \sim \text{Beta}\left(\frac{\alpha}{M}, 1\right) \quad , \quad b_m \sim \text{Beta}(\gamma, \delta), \tag{4}$$

$$s_{0m} = 0 \quad , \quad s_{tm} \sim \text{Bernoulli}(a_m^{1-s_{t-1,m}} b_m^{s_{t-1,m}}).$$

Next, we evaluate the probability of the state matrix $S$ with the transition matrix parameters $W^{(m)}$ marginalized out. We introduce the following notation, let $c_m^{00}, c_m^{01}, c_m^{10}, c_m^{11}$ be the number of $0 \to 0, 0 \to 1, 1 \to 0$ and $1 \to 1$ transitions respectively, in binary chain $m$ (including the transition from the dummy state to the first state). We can then write

$$p(S|a, b) = \prod_{m=1}^{M} (1 - a_m)^{c_m^{00}} a_m^{c_m^{01}} (1 - b_m)^{c_m^{10}} b_m^{c_m^{11}}. \tag{5}$$

We integrate out $a$ and $b$ with respect to the conjugate priors defined in equation (4) and find

$$p(S|\alpha, \gamma, \delta) = \prod_{m=1}^{M} \frac{\frac{\alpha}{M} \Gamma(\frac{\alpha}{M} + c_m^{01}) \Gamma(c_m^{00} + 1) \Gamma(\gamma + \delta) \Gamma(\delta + c_m^{10}) \Gamma(\gamma + c_m^{11})}{\Gamma(\frac{\alpha}{M} + c_m^{00} + c_m^{01} + 1) \Gamma(\gamma) \Gamma(\delta) \Gamma(\gamma + \delta + c_m^{10} + c_m^{11})}, \tag{6}$$

where $\Gamma(x)$ is the Gamma function.

## 2.2 Taking the infinite limit

Analogous to the IBP, we compute the limit for $M \to \infty$ of the finite model in equation (6). The probability of a single matrix in the limit as $M \to \infty$ is zero. This is not a problem since we are only interested in the probability of a whole class of matrices, namely those matrices that can be transformed into each other through column permutations. In other words, our factorial model is exchangeable in the columns as we don't care about the ordering of the features. Hence, we compute the infinite limit for *left-ordered form* (lof)-equivalence classes [6].

The left-ordered form of a binary $S$ matrix can be defined as follows: we interpret one column of length $T$ as encoding a binary number: column $m$ encodes the number $2^{T-1}s_{1m} + 2^{T-2}s_{2m} + \cdots + s_{Tm}$. We call the number which a feature encodes the *history* of the column. Then, we denote with $M_h$ the number of columns in the matrix $S$ that have the same history. We say a matrix is a lof-matrix if its columns are sorted in decreasing history values. Let $S$ be a lof-matrix, then we denote with $[S]$ the set of all matrices that can be transformed into $S$ using only column permutations; we call $[S]$ the lof-equivalence class. One can check that the number of elements in the lof-equivalence class of $S$ is equal to $\frac{M!}{\prod_{h=0}^{2^T-1} M_h!}$. We thus find the probability of the equivalence class of $S$ to be

$$p([S]) = \sum_{S \in [S]} p(S|\alpha, \gamma, \delta) \tag{7}$$

$$= \frac{M!}{\prod_{h=0}^{2^T-1} M_h!} \prod_{m=1}^{M} \frac{\frac{\alpha}{M} \Gamma(\frac{\alpha}{M} + c_m^{01}) \Gamma(c_m^{00} + 1) \Gamma(\gamma + \delta) \Gamma(\delta + c_m^{10}) \Gamma(\gamma + c_m^{11})}{\Gamma(\frac{\alpha}{M} + c_m^{00} + c_m^{01} + 1) \Gamma(\gamma) \Gamma(\delta) \Gamma(\gamma + \delta + c_m^{10} + c_m^{11})}. \tag{8}$$

This form allows us to compute a meaningful limit as $M \to \infty$. A writeup on the technical details of this computation can be found on the author's website. The end result has the following form

$$\lim_{M \to \infty} p([S]) = \frac{\alpha^{M_+}}{\prod_{h=0}^{2^T-1} M_h!} \exp\{-\alpha H_T\} \prod_{m=1}^{M_+} \frac{(c_m^{01} - 1)! c_m^{00}! \Gamma(\gamma + \delta) \Gamma(\delta + c_m^{10}) \Gamma(\gamma + c_m^{11})}{(c_m^{00} + c_m^{01})! \Gamma(\gamma) \Gamma(\delta) \Gamma(\gamma + \delta + c_m^{10} + c_m^{11})},$$
$$\tag{9}$$

where $H_t$ denotes the $t$'th Harmonic number and $M_+$ denotes the number of Markov chains that switch on at least once between 0 and $T$, i.e. $M_+$ is the effective dimension of our model.

## 2.3 Properties of the distribution

First of all, it is interesting to note from equation (9) that our model is exchangeable in the columns and Markov exchangeable[2] in the rows.

Next, we derive the distribution in equation (9) through a stochastic process that is analogous to the Indian Buffet Process but slightly more complicated for the actors involved. In this stochastic process, $T$ customers enter an Indian restaurant with an infinitely long buffet of dishes organized in a line. The first customer enters the restaurant and takes a serving from each dish, starting at the left of the buffet and stopping after a $\mathsf{Poisson}(\alpha)$ number of dishes as his plate becomes overburdened. A waiter stands near the buffet and takes notes as to how many people have eaten which dishes. The $t$'th customer enters the restaurant and starts at the left of the buffet. At dish $m$, he looks at the customer in front of him to see whether he has served himself that dish.

- If so, he asks the waiter how many people have previously served themselves dish $m$ when the person in front of them did (the waiters replies to him the number $c_m^{11}$) and how many people didn't serve themselves dish $m$ when the person in front of them did (the waiter replies to him the number $c_m^{10}$). The customer then serves himself dish $m$ with probability $(c_m^{11} + \delta)/(\gamma + \delta + c_m^{10} + c_m^{11})$.

- Otherwise, he asks the waiter how many people have previously served themselves dish $m$ when the person in front of them did not (the waiters replies to him the number $c_m^{01}$) and how many people didn't serve themselves dish $m$ when the person in front of them did not either (the waiter replies to him the number $c_m^{00}$). The customer then serves himself dish $m$ with probability $c_m^{00}/(c_m^{00} + c_m^{01})$.

The customer then moves on to the next dish and does exactly the same. After the customer has passed all dishes people have previously served themselves from, he tries $\mathsf{Poisson}(\alpha/t)$ new dishes. If we denote with $M_1^{(t)}$ the number of new dishes tried by the $t$'th customer, the probability of any particular matrix being produced by this process is

$$p([\boldsymbol{S}]) = \frac{\alpha^{M_+}}{\prod_{t=1}^{T} M_1^{(t)}!} \exp\{-\alpha H_T\} \prod_{m=1}^{M} \frac{\frac{\alpha}{M}\Gamma(\frac{\alpha}{M} + c_m^{01})\Gamma(c_m^{00} + 1)\Gamma(\gamma + \delta)\Gamma(\delta + c_m^{10})\Gamma(\gamma + c_m^{11})}{\Gamma(\frac{\alpha}{M} + c_m^{00} + c_m^{01} + 1)\Gamma(\gamma)\Gamma(\delta)\Gamma(\gamma + \delta + c_m^{10} + c_m^{11})}.$$
(10)

We can recover equation (9) by summing over all possible matrices that can be generated using the Markov Indian Buffet process that are in the same lof-equivalence class. It is straightforward to check that there are exactly $\frac{\prod_{t=1}^{T} M_1^{(t)}!}{\prod_{h=0}^{2^T-1} M_h!}$ of these. Multiplying this by equation (10) we recover equation (9). This construction shows that the effective dimension of the model ($M_+$) follows a $\mathsf{Poisson}(\alpha H_T)$ distribution.

## 2.4 A stick breaking representation

Although the representation above is convenient for theoretical analysis, it is not very practical for inference. Interestingly, we can adapt the stick breaking construction for the IBP [8] to the mIBP. This will be very important for the iFHMM as it will allow us to use a combination of slice sampling and dynamic programming to do inference.

The first step in the stick breaking construction is to find the distribution of $a_{(1)} > a_{(2)} > \cdots$, the order statistics of the parameters $\boldsymbol{a}$. Since the distribution on the variables $a_m$ in our model are identical to the distribution of the feature parameters in the IBP model, we can use the result in [8] that these variables have the following distribution

$$a_{(1)} \quad \propto \quad \mathsf{Beta}(\alpha, 1), \tag{11}$$
$$p(a_{(m)}|a_{(m-1)}) \quad = \quad \alpha a_{(m-1)}^{-\alpha} a_{(m)}^{\alpha-1} \mathbb{I}(0 \le a_{(m)} \le a_{(m-1)}). \tag{12}$$

The variables $b_m$ are all independent draws from a $\mathsf{Beta}(\gamma, \delta)$ distribution which is independent of $M$. Hence if we denote with $b_{(m)}$ the $b$ variable corresponding to the $m$'th largest $a$ value (in other words: the $b$ value corresponding to $a_{(m)}$) then it follows that $b_{(m)} \sim \mathsf{Beta}(\gamma, \delta)$.

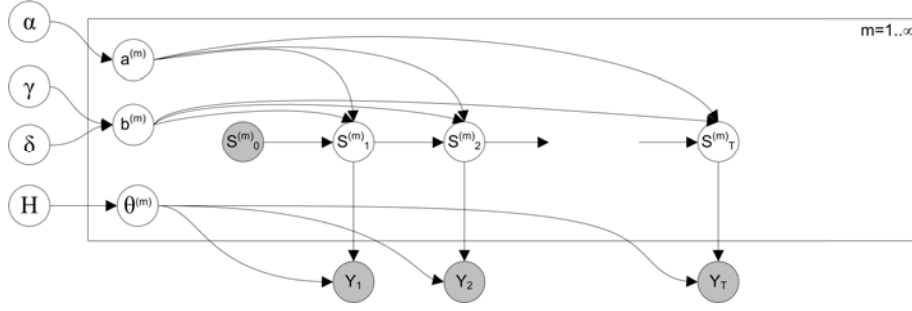

Figure 3: The Infinite Factorial Hidden Markov Model

# 3   The Infinite Factorial Hidden Markov Model

In this section, we explain how to use the mIBP as a building block in a full blown probabilistic model. The mIBP provides us with a matrix $S$ which we interpret as an arbitrarily large set of parallel Markov chains. First we augment our binary representation with a more expressive component which can describe feature specific properties. We do this by introducing a base distribution $H$ from which we sample a parameter $\theta_m \sim H$ for each Markov chain. This is a rather flexible setup as the base distribution can introduce a parameter for every chain and every timestep, which we will illustrate in section 3.1.

Now that we have a model with a more expressive latent structure, we want to add a likelihood model $F$ which describes the distribution over the observations conditional on the latent structure. Formally, $F(y_t \mid s_t, \theta)$ describes the probability of generating $y_t$ given the model parameters $\theta$ and the current latent feature state $s_t$. We note that there are two important conditions which the likelihood must satisfy in order for the limit $M \to \infty$ to be valid: (1) the likelihood must be invariant to permutations of the features, (2) the likelihood cannot depend on $\theta_m$ if $s_{tm} = 0$. Figure 3 shows the graphical model for our construction which we call the *Infinite Factorial Hidden Markov Model* (iFHMM). In the following section, we describe one particular choice of base distribution and likelihood model which performs Independent Component Analysis on time series.

## 3.1   The Independent Component Analysis iFHMM

Independent Component Analysis [9] (ICA) means different things to different people. Originally invented as an algorithm to unmix a signal into a set of independent signals, it will be more insightful for our purpose to think of ICA in terms of the probabilistic model which we describe below. As we explain in detail in section 4, we are interested in ICA to solve the blind source separation problem.

Assume that $M$ signals are represented through the vectors $\boldsymbol{x}_m$; grouping them we can represent the signals using the matrix $X = [\boldsymbol{x}_1 \boldsymbol{x}_2 \cdots \boldsymbol{x}_M]$. Next, we linearly combine the signals using a mixing matrix $W$ to generate the observed signal $Y = XW$. Additionally, we will assume IID Normal$(0, \sigma_Y^2)$ noise added: $Y = XW + \epsilon$.

A variety of fast algorithms exist which unmix the observations $Y$ and recover the signal $X$. However, crucial to these algorithms is that the number of signals is known in advance. [10] used the IBP to design the Infinite Independent Component Analysis (iICA) model which learns an appropriate number of signals from exchangeable data. Our ICA iFHMM model extends the iICA for time series.

The ICA iFHMM generative model can be described as follows: we sample $S \sim$ mIBP and pointwise multiply (denoted by $\odot$) it with a signal matrix $X$. Each entry in $X$ is an IID sample from a Laplace$(0, 1)$ distribution. One could choose many other distributions for $X$, but since in section 4 we will model speech data, which is known to be heavy tailed, the Laplace distribution is a convenient choice. Speakers will be speaking infrequently so pointwise multiplying a heavy tailed distribution with a sparse binary matrix achieves our goal of producing a sparse heavy tailed distribution. Next, we introduce a mixing matrix $W$ which has a row for each signal in $S \odot X$ and a column for each observed dimension in $Y$. The entries for $W$ are sampled IID from a Normal$(0, \sigma_W^2)$ distribution. Finally, we combine the signal and mixing matrices as in the finite case to form the

observation matrix $\boldsymbol{Y}$: $\boldsymbol{Y} = (\boldsymbol{S} \odot \boldsymbol{X})\boldsymbol{W} + \epsilon$ where $\epsilon$ is $\mathsf{Normal}(0, \sigma_Y^2)$ IID noise for each element. In terms of the general iFHMM model defined in the previous section, the base distribution $H$ is a joint distribution over columns of $\boldsymbol{X}$ and rows of $\boldsymbol{W}$. The likelihood $F$ performs the pointwise multiplication, mixes the signals and adds the noise. It can be checked that our likelihood satisfies the two technical conditions for proper iFHMM likelihoods described in section 3.

## 3.2   Inference

Inference for nonparametric models requires special treatment as the potentially unbounded dimensionality of the model makes it hard to use exact inference schemes. Traditionally, in nonparametric factor models inference is done using Gibbs sampling, sometimes augmented with Metropolis Hastings steps to improve performance. However, it is commonly known that naive Gibbs sampling in a time series model is notoriously slow due to potentially strong couplings between successive time steps [11]. In the context of the infinite hidden Markov model, a solution was recently proposed in [12], where a slice sampler adaptively truncates the infinite dimensional model after which a dynamic programming performs exact inference. Since a stick breaking construction for the iFHMM is readily available, we can use a very similar approach for the iFHMM. The central idea is the following: we introduce an auxiliary slice variable $\mu$ with the following distribution

$$\mu \sim \mathsf{Uniform}(0, \min_{m:\exists t, s_{tm}=1} a_m). \tag{13}$$

It is not essential that we sample from the uniform distribution, in fact for some of our experiments we use the more flexible Beta distribution. The resulting joint distribution is

$$p(\mu, \boldsymbol{a}, \boldsymbol{b}, \boldsymbol{S}) = p(\mu|\boldsymbol{a}, \boldsymbol{S})p(\boldsymbol{a}, \boldsymbol{b}, \boldsymbol{S}). \tag{14}$$

It is clear from the equation above that one recovers the original mIBP distribution when we integrate out $\mu$. However, when we condition the joint distribution on $\mu$ we find

$$p(\boldsymbol{S}|\boldsymbol{Y}, \mu, \boldsymbol{a}, \boldsymbol{b}) \propto p(\boldsymbol{S}|\boldsymbol{Y}, \boldsymbol{a}, \boldsymbol{b}) \frac{\mathbb{I}(0 \le \mu \le \min_{m:\exists t, s_{tm}=1} a_m)}{\min_{m:\exists t, s_{tm}=1} a_m} \tag{15}$$

which forces all columns of $\boldsymbol{S}$ for which $a_m < \mu$ to be in the all zero state. Since there can only be a finite number of $a_m > \mu$, this effectively implies that we need only resample a finite number of columns of $\boldsymbol{S}$.

We now describe our algorithm in the context of the ICA iFHMM: we start with an initial $\boldsymbol{S}$ matrix and sample $\boldsymbol{a}, \boldsymbol{b}$. Next, conditional on our initial $\boldsymbol{S}$ and the data $\boldsymbol{Y}$, we sample the ICA parameters $\boldsymbol{X}$ and $\boldsymbol{W}$. We then start an iterative sampling scheme which involves the following steps:

1. We sample the auxiliary slice variable $\mu$. This might involve extending the representation of $\boldsymbol{S}, \boldsymbol{X}$ and $\boldsymbol{W}$,

2. For all the represented features, we sample $\boldsymbol{S}, \boldsymbol{X}$ and $\boldsymbol{W}$,

3. We resample the hyperparameters $(\sigma_Y, \sigma_W, \alpha, \gamma, \delta)$ of our model,

4. We compact our representation by removing all unused features.

We experimented with 3 different algorithms for step 2. The first, a naive Gibbs sampler, did not perform well as we expected. The second algorithm, which we used for our experiments, is a blocked Gibbs sampler which fixes all but one column of $\boldsymbol{S}$ and runs a forward-filtering backward-sampling sweep on the remaining column. This allows us to analytically integrate out one column of $\boldsymbol{X}$ in the dynamic program and resample it from the posterior afterwards. $\boldsymbol{W}$ can be sampled exactly conditional on $\boldsymbol{X}, \boldsymbol{S}$ and $\boldsymbol{Y}$. A third algorithm runs dynamic programming on multiple chains at once. We originally designed this algorithm as it has the potential to merge two features in one sweep. However, we found that because we cannot integrate out $\boldsymbol{X}$ and $\boldsymbol{W}$ in this setting, the inference was not faster than our second algorithm. Note that because the bulk of the computation is used for estimating $\boldsymbol{X}$ and $\boldsymbol{W}$, the dynamic programming based algorithms are effectively as fast as the naive Gibbs sampler. A prototype implementation of the iFHMM sampler in Matlab or .NET can be obtained from the first author.

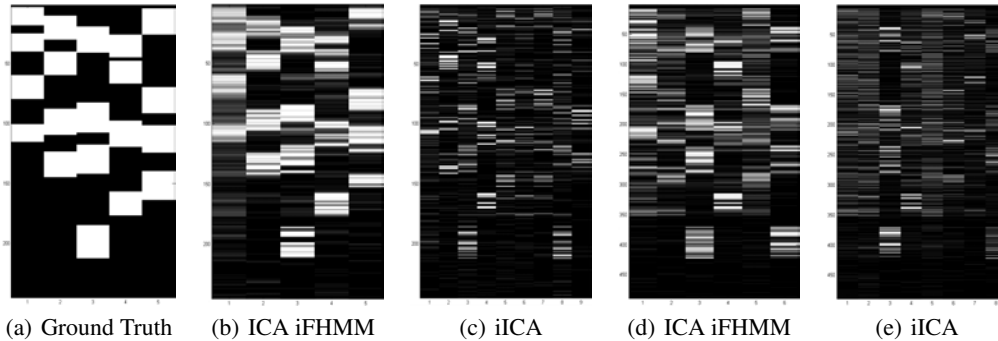

| (a) Ground Truth | (b) ICA iFHMM | (c) iICA | (d) ICA iFHMM | (e) iICA |

Figure 4: Blind speech separation experiment; figures represent which speaker is speaking at a certain point in time: columns are speakers, rows are white if the speaker is talking and black otherwise. The left figure is ground truth, the next two figures in are for the 10 microphone experiment, the right two figures are for the 3 microphone experiment.

## 4 Experiments

To test our model and inference algorithms, we address a blind speech separation task, also known as the cocktail party problem. More specifically, we record multiple people who are simultaneously speaking, using a set of microphones. Given the mixed speech signals, the goal is to separate out the individual speech signals. Key to our presentation is that we want to illustrate that using nonparametric methods, we can learn the number of speakers from a small amount of data. Our first experiment learns to recover the signals in a setting with more microphones then speakers, our second experiment uses less microphones then speakers.

The experimental setup was the following: we downloaded data from 5 speakers from the *Speech Separation Challenge* website[3]. The data for each speaker consists of 4 sentences which we appended with random pauses in between each sentence. Figure 4(a) illustrates which person is talking at what point in time. Next, we artificially mix the data 10 times. Each mixture is a linear combination of each of the 5 speakers using $\mathsf{Uniform}(0, 1)$ mixing weights. We centered the data to have zero mean and unit variance and added IID $\mathsf{Normal}(0, \sigma_Y^2)$ noise with $\sigma_Y = 0.3$.

In our first experiment we compared the ICA iFHMM with the iICA model using all 10 microphones. We subsample the data so we learn from 245 datapoints. We initialized the samplers for both models with an initial $\boldsymbol{S}$ matrix with 10 features, 5% random entries on. We use a $\mathsf{Gamma}(1.0, 4.0)$ prior on $\alpha$. In both models, we use a $\mathsf{InverseGamma}(2.0, 1.0)$ prior for $\sigma_Y$ and $\sigma_W$. Finally, for the iFHMM, we chose a $\mathsf{Gamma}(10.0, 1.0)$ prior on $\gamma$ and a $\mathsf{Gamma}(1.0, 1.0)$ prior on $\delta$ to encode our belief that people speak for larger stretches of time, say the time to pronounce a sentence. We ran the samplers for 5000 iterations and then gathered 20 samples every 20 iterations.

For both the ICA iFHMM and iICA models, we average the 20 samples and rearrange the features to have maximal overlap with the ground truth features. Figure 4(b) shows that the ICA iFHMM model recognizes that the data was generated from 5 speakers. Visual inspection of the recovered $\boldsymbol{S}$ matrix also shows that the model discovers who is speaking at what time. 4(c) illustrated the results of the iICA model on the same data. Although the model discovers some structure in the data, it fails to find the right number of speakers (it finds 9) and does a poor job in discovering which speaker is active at which time. We computed the average mutual information between the 5 columns of the true $\boldsymbol{S}$ matrix and the first 5 columns of the recovered $\boldsymbol{S}$ matrices. We find that the iFHMM has an average mutual information of $0.296$ compared to $0.068$ for the iICA model. The difference between the two models is strictly limited to the difference between using the IBP versus mIBP. We want to emphasize that although one could come up with ad-hoc heuristics to smooth the iICA results, the ICA iFHMM is a principled probabilistic model that does a good job at comparable computational cost.

In a second experiment, we chose to perform blind speech separation using only the first 3 microphones. We subsampled a noiseless version of the data to get 489 datapoints. We ran both the ICA iFHMM and iICA inference algorithms using exactly the same settings as in the previous experi-

ment. Figure 4(d) and 4(e) show the average of 20 samples, rearranged to match the ground truth. In this setting both methods fail to identify the number of speakers although the ICA iFHMM clearly performs better. The ICA iFHMM finds one too many signal: the spurious signal is very similar to the third signal which suggests that the error is a problem of the inference algorithm and not so much of the model itself. The iICA on the other hand performs poorly: it is very hard to find any structure in the recovered Z matrix. We compared the mutual information as described above and find that the iFHMM has a mutual information of 0.091 compared to 0.028 for the iICA model.

## 5   Discussion

The success of the Hidden Markov Model set off a wealth of extensions to adapt it to particular situations. [2] introduced a *factorial* hidden Markov model which explicitly models dynamic latent features while in [13] a *nonparametric* version of the the Hidden Markov Model was presented. In this paper we "complete the square" by presenting a nonparametric Factorial Hidden Markov Model. We introduced a new stochastic process for latent feature representation of time series called the Markov Indian Buffet Process. We showed how this stochastic process can be used to build a nonparametric extension of the FHMM which we call the iFHMM. Another issue which deserves further exploration is inference: in [2] it was found that a structured variational method provides a good balance between accuracy and computational effort. An interesting open problem is whether we can adapt the structured variational method to the iFHMM. Finally, analogous to the two-parameter IBP [14] we would like to add one more degree of flexibility to control the $0 \rightarrow 1$ transition probability more finely. Although the derivation of the mIBP with this extra parameter is straightforward, we as yet lack a stick breaking construction for this model which is crucial for our inference scheme.

**Acknowledgments**

We kindly acknowledge David Knowles for discussing the generalized Amari error and A. Taylan Cemgil for his suggestions on blind source separation. Jurgen Van Gael is supported by a Microsoft Research PhD scholarship; Zoubin Ghahramani is also in the Machine Learning department, CMU.

**References**

[1] L. R. Rabiner, "A tutorial on hidden markov models and selected applications in speech recognition," *Proceedings of the IEEE*, vol. 77, pp. 257–286, 1989.

[2] Z. Ghahramani and M. I. Jordan, "Factorial hidden markov models," *Machine Learning*, vol. 29, pp. 245–273, 1997.

[3] P. Wang and Q. Ji, "Multi-view face tracking with factorial and switching hmm," in *Proceedings of the Seventh IEEE Workshops on Application of Computer Vision*, pp. 401–406, IEEE Computer Society, 2005.

[4] B. Logan and P. Moreno, "Factorial hmms for acoustic modeling," 1998.

[5] K. Duh, "Joint labeling of multiple sequences: A factorial hmm approach," in *43rd Annual Meeting of the Association of Computational Linguistics (ACL) - Student Research Workshop*, 2005.

[6] T. L. Griffiths and Z. Ghahramani, "Infinite latent feature models and the indian buffet process," *Advances in Neural Information Processing Systems*, vol. 18, pp. 475–482, 2006.

[7] R. M. Neal, "Bayesian mixture modeling," *Maximum Entropy and Bayesian Methods*, 1992.

[8] Y. W. Teh, D. Görür, and Z. Ghahramani, "Stick-breaking construction for the indian buffet process," *Proceedings of the International Conference on Artificial Intelligence and Statistics*, vol. 11, 2007.

[9] A. Hyvarinen and E. Oja, "Independent component analysis: Algorithms and applications," *Neural Networks*, vol. 13, pp. 411–30, 2000.

[10] D. Knowles and Z. Ghahramani, "Infinite sparse factor analysis and infinite independent components analysis," *Lecture Notes in Computer Science*, vol. 4666, p. 381, 2007.

[11] S. L. Scott, "Bayesian methods for hidden markov models: Recursive computing in the 21st century," *Journal of the American Statistical Association*, vol. 97, pp. 337–351, Mar. 2002.

[12] J. Van Gael, Y. Saatci, Y. W. Teh, and Z. Ghahramani, "Beam sampling for the infinite hidden markov model," in *The 25th International Conference on Machine Learning*, vol. 25, (Helsinki), 2008.

[13] M. J. Beal, Z. Ghahramani, and C. E. Rasmussen, "The infinite hidden markov model," *Advances in Neural Information Processing Systems*, vol. 14, pp. 577 – 584, 2002.

[14] Z. Ghahramani, T. L. Griffiths, and P. Sollich, "Bayesian nonparametric latent feature models," *Bayesian Statistics*, vol. 8, 2007.

## Footnotes

[2]A sequence is Markov exchangeable if its distribution is invariant under permutations of the transitions.

[3]http://www.dcs.shef.ac.uk/ martin/SpeechSeparationChallenge.htm
